# Active Learning for Anomaly and Rare-Category Detection

**Dan Pelleg and Andrew Moore**
School of Computer Science
Carnegie-Mellon University
Pittsburgh, PA 15213 USA
`dpelleg@cs.cmu.edu, awm@cs.cmu.edu`

## Abstract

We introduce a novel active-learning scenario in which a user wants to work with a learning algorithm to identify *useful* anomalies. These are distinguished from the traditional statistical definition of anomalies as outliers or merely ill-modeled points. Our distinction is that the usefulness of anomalies is categorized subjectively by the user. We make two additional assumptions. First, there exist extremely few useful anomalies to be hunted down within a massive dataset. Second, both useful and useless anomalies may sometimes exist within tiny classes of similar anomalies. The challenge is thus to identify "rare category" records in an unlabeled noisy set with help (in the form of class labels) from a human expert who has a small budget of datapoints that they are prepared to categorize. We propose a technique to meet this challenge, which assumes a mixture model fit to the data, but otherwise makes no assumptions on the particular form of the mixture components. This property promises wide applicability in real-life scenarios and for various statistical models. We give an overview of several alternative methods, highlighting their strengths and weaknesses, and conclude with a detailed empirical analysis. We show that our method can quickly zoom in on an anomaly set containing a few tens of points in a dataset of hundreds of thousands.

## 1   Introduction

We begin with an example of a rare-category-detection problem: an astronomer needs to sift through a large set of sky survey images, each of which comes with many numerical parameters. Most of the objects (99.9%) are well explained by current theories and models. The remainder are anomalies, but 99% of these anomalies are uninteresting, and only 1% of them (0.001% of the full dataset) are useful. The first type of anomalies, called "boring anomalies", are records which are strange for uninteresting reasons such as sensor faults or problems in the image processing software. The useful anomalies are extraordinary objects which are worthy of further research. For example, an astronomer might want to cross-check them in various databases and allocate telescope time to observe them in greater detail. The goal of our work is finding this set of rare and useful anomalies.

Although our example concerns astrophysics, this scenario is a promising general area for exploration wherever there is a very large amount of scientific, medical, business or intelligence data and a domain expert wants to find truly exotic rare events while not becoming

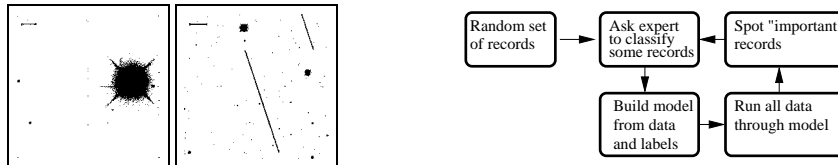

Figure 1: Anomalies in Sloan data: Diffraction spikes (left). Satellite trails (center). The active-learning loop is shown on the right.

swamped with uninteresting anomalies. Two rare categories of "boring" anomalies in our test astrophysics data are shown in Figure 1. The first, a well-known optical artifact, is the phenomenon of diffraction spikes. The second consists of satellites that happened to be flying overhead as the photo was taken.

As a first step, we might try defining a statistical model for the data, and identifying objects which do not fit it well. At this point, objects flagged as "anomalous" can still be almost entirely of the uninteresting class of anomalies. The computational and statistical question is then how to use feedback from the human user to iteratively reorder the queue of anomalies to be shown to the user in order to increase the chance that the user will soon see an anomaly of a whole new category.

We do this in the familiar pool-based active learning framework[1]. In our setting, learning proceeds in rounds. Each round starts with the teacher labeling a small number of examples. Then the learner models the data, taking into account the labeled examples as well as the remainder of the data, which we assume to be much larger in volume. The learner then identifies a small number of input records ("hints") which are important in the sense that obtaining labels for them would help it improve the model. These are shown to the teacher (in our scenario, a human expert) for labeling, and the cycle repeats. The model, which we call "irrelevance feedback", is shown in Figure 1.

It may seem too demanding to ask the human expert to give class labels instead of a simple "interesting" or "boring" flag. But in practice, this is not an issue—it seems easier to place objects into such "mental bins". For example, in the astronomical data we have seen a user place most objects into previously-known categories: point sources, low-surface-brightness galaxies, etc. This also holds for the negative examples: it is frustrating to have to label all anomalies as "bad" without being able to explain why. Often, the data is better understood as time goes by, and users wish to revise their old labels in light of new examples. Note that the statistical model does not care about the names of the labels. For all it cares, the label set can be utterly changed by the user from one round to another. Our tools allow that: the labels are unconstrained and the user can add, refine, and delete classes at will. It is trivial to accommodate the simpler "interesting or not" model in this richer framework.

Our work differs from traditional applications of active learning in that we assume the distribution of class sizes to be extremely skewed. For example, the smallest class may have just a few members whereas the largest may contain a few million. Generally in active learning, it is believed that, right from the start, examples from each class need to be presented to the oracle [1, 2, 3]. If the class frequencies were balanced, this could be achieved by random sampling. But in datasets with the rare categories property, this no longer holds, and much of our effort is an attempt to remedy the situation.

Previous active-learning work tends to tie intimately to a particular model [4, 3]. We would like to be able to "plug in" different types of models or components and therefore propose model-independent criteria. The same reasoning also precludes us from directly using distances between data points, as is done in [5].

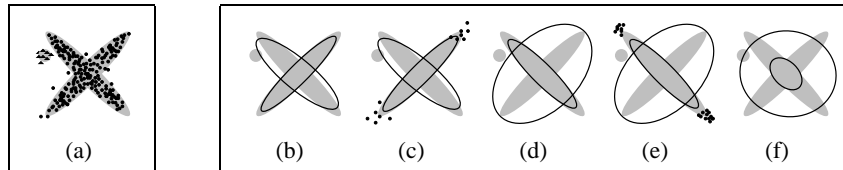

Figure 2: Underlying data distribution for the example (a); behavior of the lowlik method (b–f). The original data distribution is in (a). The unsupervised model fit to it in (b). The anomalous points according to lowlik, given the model in (b), are shown in (c). Given labels for the points in (c), the model in (d) is fitted. Given the new model, anomalous points according to lowlik are flagged (e). Given labels for the points in (c) and (e), this is the new fitted model (f).

Another desired property is resilience to noise. Noise can be inherent in the data (e.g., from measurement errors) or be an artifact of a ill-fitting model. In any case, we need to be able to identify query points in the presence of noise. This is a not just a bonus feature: points which the model considers noisy could very well be the key to improvement if presented to the oracle. This is in contrast to the approach taken by some: a pre-assumption that the data is noiseless [6, 7].

## 2 Overview of Hint Selection Methods

In this section we survey several proposed methods for active learning as they apply to our setting. While the general tone is negative, what follows should not be construed as general dismissal of these methods. Rather, it is meant to highlight specific problems with them when applied to a particular setting. Specifically, the rare-categories assumption (and in some cases, just having more than 2 classes) breaks the premises for some of them.

As an example, consider the data shown in Figure 2 (a). It is a mixture of two classes. One is an X-shaped distribution, from which 2000 points are drawn. The other is a circle with 100 points. In this example, the classifier is a Gaussian Bayes classifier trained in a semi-supervised manner from labeled and unlabeled data, with one Gaussian per class. The model is learned with a standard EM procedure, with the following straightforward modification [8, 9] to enable semi-supervised learning. Before each M step we clamp the class membership values for the hinted records to match the hints (i.e., one for the labeled class for this record, and zero elsewhere).

Given fully labeled data, our learner would perfectly predict class membership for this data (although it would be a poor generative model): one Gaussian centered on the circle, and another spherical Gaussian with high variance centered on the X. Now, suppose we plan to perform active learning in which we take the following steps:

1. Start with entirely unlabeled data.
2. Perform semi-supervised learning (which, on the first iteration degenerates to unsupervised learning).
3. Ask an expert to classify the 35 strangest records.
4. Go to Step 2.

On the first iteration (when unsupervised) the algorithm will naturally use the two Gaussians to model the data as in Figure 2(b), with one Gaussian for each of the arms of the "X", and the points in the circle represented as members of one of them. What happens next all depends on the choice of the datapoints to show to the human expert. We now survey the methods for hint selection.

**Choosing Points with Low Likelihood:** A rather intuitive approach is to select as hints the points which the model performs worst on. This can be viewed as model variance

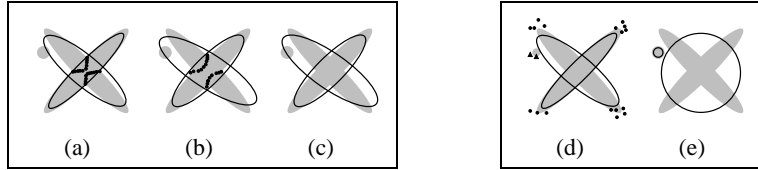

Figure 3: Behavior of the ambig (a–c) and interleave (d–e) methods. The unsupervised model and the points which ambig flags as anomalous, given this model (a). The model learned using labels for these points is (b), along with the point *it* flags. The last refinement, given both sets of labels (c).

minimization [4] or as selection of points furthest away from any labeled points [5]. We do this by ranking each point in order of increasing model likelihood, and choosing the most anomalous items.

We show what this approach would flag in the given configuration in Figure 2. It is derived from a screenshot of a running version of our code, redrawn by hand for clarity. Each subsequent drawing shows a model which EM converged to after including the new labels, and the hints it chooses under a particular scheme (here it is what we call lowlik). These hints affect the model shown for the next round. The underlying distribution is shown in gray shading. We use this same convention for the other methods below.

In the first round, the Mahalanobis distance for the points in the corners is greater than those in the circle, therefore they are flagged. Another effect we see is that one of the arms is represented more heavily. This is probably due to its lower variance. In any event, none of the points in the circle is flagged. The outcome is that the next round ends up in a similar local minimum. We can also see that another step will not result in the desired model. Only after obtaining labels for all of the "outlier" points (that is, those on the extremes of the distribution) will this approach go far enough down the list to hit a point in the circle. This means that in scenarios where there are more than a few hundred noisy data, classification accuracy is likely to be very low.

**Choosing Ambiguous Points:** Another popular approach is to choose the points which the learner is least certain about. This is the spirit of "query by committee" [10] and "uncertainty sampling" [11]. In our setting this is implemented in the following way. For each data point, the EM algorithm maintains an estimate of the probability of its membership in every mixture component. For each point, we compute the entropy of the set of all such probabilities, and rank the points in decreasing order of the entropy. This way, the top of the list will have the objects which are "owned" by multiple components.

For our example, this would choose the points shown in Figure 3. As expected, points on the decision boundaries between classes are chosen. Here, the ambiguity sets are useless for the purpose of modeling the entire distribution. One might argue this only holds for this contrived distribution. However, in general this is a fairly common occurrence, in the sense that the ambiguity criterion works to nudge the decision surfaces so they better fit a relatively small set of labeled examples. It may help modeling the points very close to the boundaries, but it does not improve generalization accuracy in the general case. Indeed, we see that if we repeatedly apply this criterion we end up asking for labels for a great number of points in close proximity, to very little effect on the overall model. In the results section below, we call this method ambig.

**Combining Unlikely and Ambiguous Points:** Our next candidate is a hybrid method which tries to combine the hints from the two previous methods. Recall they both produce a ranked list of all the points. We merge the lists into another ranked list in the following way. Alternate between the lists when picking items. For each list, pick the top item that has not already been placed in the output list. When all elements are taken, the output list is a ranked list as required. We now pick the top items from this list for hints.

As expected we get a good mix of points in both hint sets (not shown). But, since neither method identifies the small cluster, their union fails to find it as well. However, in general it is useful to combine different criteria in this way, as our empirical results below show. There, this method is called mix-ambig-lowlik.

**Interleaving:** We now present what we consider is the logical conclusion of the observations above. To the best of our knowledge, the approach is novel. The key insight is that our group of anomalies was, in fact, reasonably ordinary when analyzed on a *global* scale. In other words, the mixture density of the region we chose for the group of anomalies is not sufficiently low for them to rank high on the hint list. Recall that the mixture model sums up the weighted per-model densities. Therefore, a point that is "split" among several components approximately evenly, and scores reasonably high on at least some of them, will not be flagged as anomalous.

Another instance of the same problem occurs when a point which is somewhat "owned" by a component with high mixture weight. Even if the small component that "owns" most of it predicts it is very unlikely, that term has very little effect on the overall density.

Therefore, our goal is to eliminate the mixture weights from the equation. Our idea is that if we restrict the focus to match the "point of view" of just one component, these anomalies will become more apparent. We do this by considering just the points that "belong" to one component, and by ranking them according to the PDF of this component. The hope is that given this restricted view, anomalies that do not fit the component's own model will stand out.

More precisely, let $c$ be a component and $i$ a data point. The EM algorithm maintains, for every $c$ and $i$, an estimate $z_i^c$ of the degree of "ownership" that $c$ exerts over $i$. For each component $c$ we create a list of all the points for which $c = \arg\max_{c'} z_i^{c'}$, ranked by $z_i^c$.

Having constructed the sorted lists, we merge them in a generalization of the merge method described above. We cycle through the lists in some order. For each list, we pick the top item that has not already been placed in the output list, and place it at the next position in the output list.

This strategy is appealing intuitively, although we have no further theoretical justification for it. We show results for this strategy for our example in Figure 3, and in the experimental section below. We see it meets the requirement of representation for all true components. Most of the points are along the major axes of the two elongated Gaussians, but two of the points are inside the small circle. Correct labels for even just these two points result in perfect classification in the next EM run.

In our experiments, we found it beneficial to modify this method as follows. One of the components is a uniform-density "background". This modification lets it nominate hints more often than any other component. In terms of list merging, we take one element from each of the lists of standard components, and then several elements from the list produced for the background component. All of the results shown were obtained using an oversampling ratio of 20. In other words, if there are $N$ components (excluding uniform), then the first cycle of hint nomination will result in $20 + N$ hints, 20 of which from uniform.

## 3   Experimental Results

To establish the results hinted by the intuition above, we conducted a series of experiments. The first one uses synthetic data. The data distribution is a mixture of components in $5, 10, 15$ and $20$ dimensions. The class size distribution is a geometric series with the largest class owning half of the data and each subsequent class being half as small.

The components are multivariate Gaussians whose covariance structure can be modeled

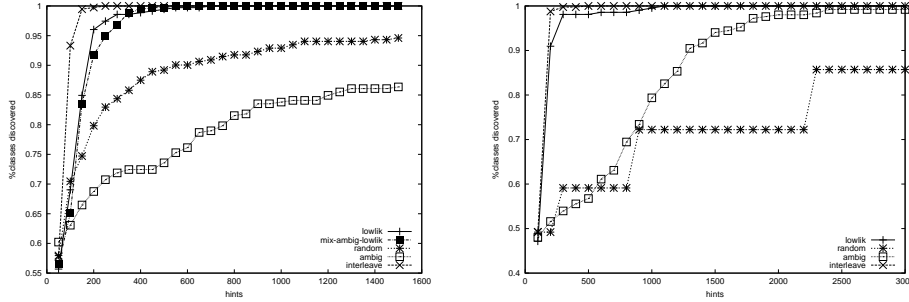

Figure 4: Learning curves for simulated data drawn from a mixture of dependency trees (left), and for the SHUTTLE set (right). The $Y$ axis shows the fraction of classes represented in queries sent to the teacher. For SHUTTLE and ABALONE below, mix-ambig-loglike is omitted because it is so similar to lowlik.

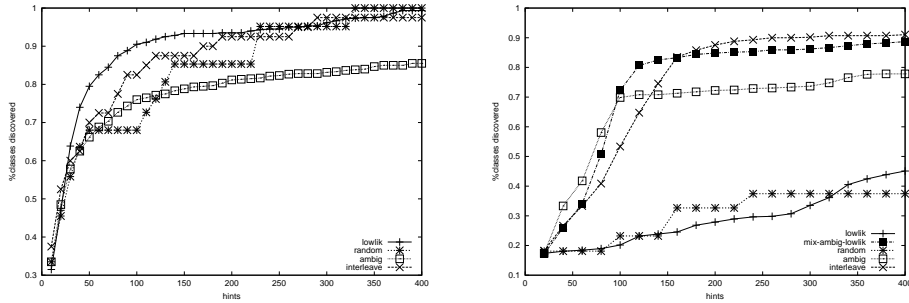

Figure 5: Learning curves for the ABALONE (left) and KDD (right) sets.

with dependency trees. Each Gaussian component has its covariance generated in the following way. Random attribute pairs are chosen, and added to an undirected dependency tree structure unless they close a cycle. Each edge describes a linear dependency between nodes, with the coefficients drawn uniformly at random, with random noise added to each value. Each data set contains $10,000$ points. There are ten tree classes and a uniform background component. The number of "background" points ranges from $50$ to $200$. Only the results for $15$ dimensions and $100$ noisy points are shown as they are representative of the other experiments. In each round of learning, the learner queries the teacher with a list of $50$ points for labeling, and has access to all the queries and replies submitted previously.

This data generation scheme is still very close to the one which our tested model assumes. Note, however, that we do not require different components to be easily identifiable. The results of this experiment are shown in Figure 4. Also included, are results for random, which is a baseline method choosing hints at random.

Our scoring function is driven by our application, and estimates the amount of effort the teacher has to expend before being presented by representatives of every single class. The assumption is that the teacher can generalize from a single example (or a very few examples) to an entire class, and the valuable information is concentrated in the first queried member of each class. More precisely, if there are $n$ classes, then the score under this metric is $1/n$ times the number of classes represented in the query set. In the query set we include all items queried in preceding rounds, as we do for other applicable metrics.

The best performer so far is interleave, taking five rounds or less to reveal all of the classes, including the very rare ones. Below we show it is superior in many of the real-life data sets. We can also see that ambig performs worse than random. This can be explained by the fact that ambig only chooses points that already have several existing components "competing" for them. Rarely do these points belong to a new, yet-undiscovered component.

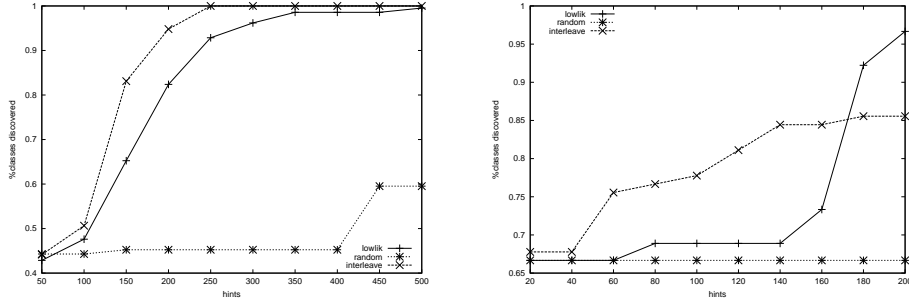

Figure 6: Learning curves for the EDSGC (left) and SDSS (right) sets.

Table 1: Properties of the data sets used.

| NAME | DIMS | RECORDS | CLASSES | SMALLEST CLASS | LARGEST CLASS | SOURCE |
|---|---|---|---|---|---|---|
| SHUTTLE | 9 | 43500 | 7 | 0.01% | 78.4% | [12] |
| ABALONE | 7 | 4177 | 20 | 0.34% | 16% | [13] |
| KDD | 33 | 50000 | 19 | 0.002% | 21.6% | [13] |
| EDSGC | 26 | 1439526 | 7 | 0.002% | 76% | [14] |
| SDSS | 22 | 517371 | 3 | 0.05% | 50.6% | [15] |

We were concerned that the poor performance of lowlik was just a consequence of our choice of metric. After all, it does not measure the number of noise points (i.e points from the uniform background component) found. These points are genuine anomalies, so it is possible that lowlik is being penalized unfairly for its focusing on the noise points. After examining the fraction of noise points (i.e., points drawn from the uniform background component) found by each algorithm, we discovered that lowlik actually scores worse than interleave on this metric as well.

The remaining experiments were run on various real data sets. Table 1 has a summary of their properties. They represent data and computational effort orders of magnitude larger than any active-learning result of which we are aware.

Results for the SHUTTLE set appear in Figure 4. We see that it takes the interleave algorithm five rounds to spot all classes, whereas the next best is lowlik, with 11. The ABALONE set (Figure 5) is a very noisy set, in which random seems to be the best long-term strategy. Again, note how ambig performs very poorly.

Due to resource limitations, results for kdd were obtained on a 50000-record random subsample of the original training set (which is roughly ten times bigger). This set has an extremely skewed distribution of class sizes, and a large number of classes. In Figure 5 we see that lowlik performs uncharacteristically poorly. Another surprise is that the combination of lowlik and ambig outperforms them both. It also matches interleave in performance, and this is the only case where we have seen it do so.

The EDSGC set, as distributed, is unlabeled. The class labels relate to the shape and size of the sky object. We see in Figure 6 that for the purpose of class discovery, we can do a good job in a small number of rounds: here, a human would have had to label just 250 objects before being presented with a member of the smallest class - comprising just 24 records out of a set of 1.4 million.

## 4   Conclusion

We have shown that some of the popular methods for active learning perform poorly in realistic active-learning scenarios where classes are imbalanced. Working from the definition of a mixture model we were able to propose methods which let each component "nominate" its favorite queries. These methods work well in the presence of noisy data and extremely rare classes and anomalies. Our simulations show that a human user only

needs to label one or two hundred examples before being presented with very rare anomalies in huge data sets. In our experience, this kind of interaction takes just an hour or two of combined human and computer time [16].

We make no assumptions about the particular form a component takes. Consequently, we expect our results to apply to many different kinds of component models, including the case where components are not dependency trees, or even not all from the same distribution.

We are using lessons learned from our empirical comparison in an application for anomaly-hunting in the astrophysics domain. Our application presents multiple indicators to help a user spot anomalous data, as well as controls for labeling points and adding classes. The application will be described in a companion paper.

## Footnotes

[1]More precisely, we allow multiple queries and labels in each learning round — the traditional presentation has just one.

## References

[1] Sugato Basu, Arindam Banerjee, and Raymond J. Mooney. Active semi-supervision for pairwise constrained clustering. Submitted for publication, February, 2003.

[2] M. Seeger. Learning with labeled and unlabeled data. Technical report, Institue for Adaptive and Neural Computation, Universiy of Edinburgh, 2000.

[3] Klaus Brinker. Incorporating diversity in active learning with support vector machines. In *Proceedings of the Twentieth International Conference on Machine Learning*, 2003.

[4] David A. Cohn, Zoubin Ghahramani, and Michael I. Jordan. Active learning with statistical models. In G. Tesauro, D. Touretzky, and T. Leen, editors, *Advances in Neural Information Processing Systems*, volume 7, pages 705–712. The MIT Press, 1995.

[5] Nirmalie Wiratunga, Susan Craw, and Stewart Massie. Index driven selective sampling for CBR, 2003. To appear in Proceedings of the Fifth International Conference on Case-Based Reasoning, Springer-Verlag, Trondheim, Norway, 23-26 June 2003.

[6] David Cohn, Les Atlas, and Richard Ladner. Improving generalization with active learning. *Machine Learning*, 15(2):201–221, 1994.

[7] Mark Plutowski and Halbert White. Selecting concise training sets from clean data. *IEEE Transactions on Neural Networks*, 4(2):305–318, March 1993.

[8] Shahshashani and Landgrebe. The effect of unlabeled examples in reducing the small sample size problem. *IEEE Trans Geoscience and Remote Sensing*, 32(5):1087–1095, 1994.

[9] Miller and Uyar. A mixture of experts classifier with learning based on both labeled and unlabelled data. In *NIPS-9*, 1997.

[10] H. S. Seung, Manfred Opper, and Haim Sompolinsky. Query by committee. In *Computational Learning Theory*, pages 287–294, 1992.

[11] David D. Lewis and Jason Catlett. Heterogeneous uncertainty sampling for supervised learning. In William W. Cohen and Haym Hirsh, editors, *Proceedings of ICML-94, 11th International Conference on Machine Learning*, pages 148–156, New Brunswick, US, 1994. Morgan Kaufmann Publishers, San Francisco, US.

[12] P.Brazdil and J.Gama. StatLog, 1991. `http://www.liacc.up.pt/ML/statlog`.

[13] C.L. Blake and C.J. Merz. UCI repository of machine learning databases, 1998. `http://www.ics.uci.edu/~mlearn/MLRepository.html`.

[14] R. C. Nichol, C. A. Collins, and S. L. Lumsden. The Edinburgh/Durham southern galaxy catalogue — IX. Submitted to the Astrophysical Journal, 2000.

[15] SDSS. *The Sloan Digital Sky Survey*, 1998. `www.sdss.org`.

[16] Dan Pelleg. *Scalable and Practical Probability Density Estimators for Scientific Anomaly Detection*. PhD thesis, Carnegie-Mellon University, 2004. Tech Report CMU-CS-04-134.

[17] David MacKay. Information-based objective functions for active data selection. *Neural Computation*, 4(4):590–604, 1992.

[18] Fabio Gagliardi Cozman, Ira Cohen, and Marclo Cesar Cirelo. Semi-supervised learning of mixture models and bayesian networks. In *Proceedings of the Twentieth International Conference on Machine Learning*, 2003.

[19] Yoram Baram, Ran El-Yaniv, and Kobi Luz. Online choice of active learning algorithms. In *Proceedings of the Twentieth International Conference on Machine Learning*, 2003.

[20] Sanjoy Dasgupta. Analysis of a greedy active learning strategy. In *Advances in Neural Information Processing Systems 18*, 2004.
